# Robustness in Markov Decision Problems with Uncertain Transition Matrices*

**Arnab Nilim**
Department of EECS [†]
University of California
Berkeley, CA 94720
*nilim@eecs.berkeley.edu*

**Laurent El Ghaoui**
Department of EECS
University of California
Berkeley, CA 94720
*elghaoui@eecs.berkeley.edu*

## Abstract

Optimal solutions to Markov Decision Problems (MDPs) are very sensitive with respect to the state transition probabilities. In many practical problems, the estimation of those probabilities is far from accurate. Hence, estimation errors are limiting factors in applying MDPs to real-world problems. We propose an algorithm for solving finite-state and finite-action MDPs, where the solution is guaranteed to be robust with respect to estimation errors on the state transition probabilities. Our algorithm involves a statistically accurate yet numerically efficient representation of uncertainty, via Kullback-Leibler divergence bounds. The worst-case complexity of the robust algorithm is the same as the original Bellman recursion. Hence, robustness can be added at practically no extra computing cost.

## 1 Introduction

We consider a finite-state and finite-action Markov decision problem in which the transition probabilities themselves are uncertain, and seek a robust decision for it. Our work is motivated by the fact that in many practical problems, the transition matrices have to be estimated from data. This may be a difficult task and the estimation errors may have a huge impact on the solution, which is often quite sensitive to changes in the transition probabilities [3]. A number of authors have addressed the issue of uncertainty in the transition matrices of an MDP. A Bayesian approach such as described by [9] requires a perfect knowledge of the whole prior distribution on the transition matrix, making it difficult to apply in practice. Other authors have considered the transition matrix to lie in a given set, most typically a polytope: see [8, 10, 5]. Although our approach allows to describe the uncertainty on the transition matrix by a polytope, we may argue *against* choosing such a model for the uncertainty. First, a general polytope is often not a tractable way to address the robustness problem, as it incurs a significant additional computational effort to handle uncertainty. Perhaps more importantly, polytopic models, especially interval matrices, may be very poor representations of statistical uncertainty and lead to very conservative robust

[†]Electrical Engineering and Computer Sciences

policies. In [1], the authors consider a problem dual to ours, and provide a general statement according to which the cost of solving their problem is polynomial in problem size, provided the uncertainty on the transition matrices is described by convex sets, without proposing any specific algorithm. This paper is a short version of a longer report [2], which contains all the proofs of the results summarized here.

**Notation.** $P > 0$ or $P \geq 0$ refers to the strict or non-strict componentwise inequality for matrices or vectors. For a vector $p > 0$, $\log p$ refers to the componentwise operation. The notation $\mathbf{1}$ refers to the vector of ones, with size determined from context. The probability simplex in $\mathbf{R}^n$ is denoted $\Delta_n = \{p \in \mathbf{R}^n_+ : p^T \mathbf{1} = 1\}$, while $\Theta_n$ is the set of $n \times n$ transition matrices (componentwise non-negative matrices with rows summing to one). We use $\sigma_{\mathcal{P}}$ to denote the support function of a set $\mathcal{P} \subseteq \mathbf{R}^n$, with for $v \in \mathbf{R}^n$, $\sigma_{\mathcal{P}}(v) := \sup\{p^T v : p \in \mathcal{P}\}$.

## 2 The problem description

We consider a finite horizon Markov decision process with finite decision horizon $T = \{0, 1, 2, \ldots, N-1\}$. At each stage, the system occupies a state $i \in \mathcal{X}$, where $n = |\mathcal{X}|$ is finite, and a decision maker is allowed to choose an action $a$ deterministically from a finite set of allowable actions $\mathcal{A} = \{a_1, \ldots, a_m\}$ (for notational simplicity we assume that $\mathcal{A}$ is not state-dependent). The system starts in a given initial state $i_0$. The states make Markov transitions according to a collection of (possibly time-dependent) transition matrices $\tau := (P_t^a)_{a \in \mathcal{A}, t \in T}$, where for every $a \in \mathcal{A}$, $t \in T$, the $n \times n$ transition matrix $P_t^a$ contains the probabilities of transition under action $a$ at stage $t$. We denote by $\pi = (\mathbf{a}_0, \ldots, \mathbf{a}_{N-1})$ a generic controller policy, where $\mathbf{a}_t(i)$ denotes the controller action when the system is in state $i \in \mathcal{X}$ at time $t \in T$. Let $\Pi = \mathcal{A}^{nN}$ be the corresponding strategy space. Define by $c_t(i, a)$ the cost corresponding to state $i \in \mathcal{X}$ and action $a \in \mathcal{A}$ at time $t \in T$, and by $c_N$ the cost function at the terminal stage. We assume that $c_t(i, a)$ is non-negative and finite for every $i \in \mathcal{X}$ and $a \in \mathcal{A}$.

For a given set of transition matrices $\tau$, we define the finite-horizon *nominal* problem by

$$\phi_N(\Pi, \tau) := \min_{\pi \in \Pi} C_N(\pi, \tau), \tag{1}$$

where $C_N(\pi, \tau)$ denotes the *expected total cost* under controller policy $\pi$ and transitions $\tau$:

$$C_N(\pi, \tau) := \mathbf{E}\left(\sum_{t=0}^{N-1} c_t(i_t, \mathbf{a}_t(i)) + c_N(i_N)\right). \tag{2}$$

A special case of interest is when the expected total cost function bears the form (2), where the terminal cost is zero, and $c_t(i, a) = \nu^t c(i, a)$, with $c(i, a)$ now a constant cost function, which we assume non-negative and finite everywhere, and $\nu \in (0, 1)$ is a discount factor. We refer to this cost function as the discounted cost function, and denote by $C_\infty(\pi, \tau)$ the limit of the discounted cost (2) as $N \to \infty$.

When the transition matrices are exactly known, the corresponding nominal problem can be solved via a dynamic programming algorithm, which has total complexity of $nmN$ flops in the finite-horizon case. In the infinite-horizon case with a discounted cost function, the cost of computing an $\epsilon$-suboptimal policy via the Bellman recursion is $O(nm \log(1/\epsilon))$; see [7] for more details.

### 2.1 The robust control problems

At first we assume that when for each action $a$ and time $t$, the corresponding transition matrix $P_t^a$ is only known to lie in some given subset $\mathcal{P}^a$. Two models for transition matrix uncertainty are possible, leading to two possible forms of finite-horizon robust control

problems. In a first model, referred to as the *stationary uncertainty* model, the transition matrices are chosen by nature depending on the controller policy once and for all, and remain fixed thereafter. In a second model, which we refer to as the *time-varying uncertainty* model, the transition matrices can vary arbitrarily with time, within their prescribed bounds. Each problem leads to a game between the controller and nature, where the controller seeks to minimize the maximum expected cost, with nature being the maximizing player.

Let us define our two problems more formally. A *policy of nature* refers to a specific collection of time-dependent transition matrices $\tau = (P_t^a)_{a \in \mathcal{A}, t \in T}$ chosen by nature, and the set of admissible policies of nature is $\mathcal{T} := (\otimes_{a \in \mathcal{A}} \mathcal{P}^a)^N$. Denote by $\mathcal{T}_s$ the set of stationary admissible policies of nature:

$$\mathcal{T}_s = \left\{ \tau = (P_t^a)_{a \in \mathcal{A}, t \in T} \in \mathcal{T} \; : \; P_t^a = P_s^a \text{ for every } t, s \in T, \; a \in \mathcal{A} \right\}.$$

The stationary uncertainty model leads to the problem

$$\phi_N(\Pi, \mathcal{T}_s) := \min_{\pi \in \Pi} \max_{\tau \in \mathcal{T}_s} C_N(\pi, \tau). \tag{3}$$

In contrast, the time-varying uncertainty model leads to a relaxed version of the above:

$$\phi_N(\Pi, \mathcal{T}_s) \le \phi_N(\Pi, \mathcal{T}) := \min_{\pi \in \Pi} \max_{\tau \in \mathcal{T}} C_N(\pi, \tau). \tag{4}$$

The first model is attractive for statistical reasons, as it is much easier to develop statistically accurate sets of confidence when the underlying process is time-invariant. Unfortunately, the resulting game (3) seems to be hard to solve. The second model is attractive as one can solve the corresponding game (4) using a variant of the dynamic programming algorithm seen later, but we are left with a difficult task, that of estimating a meaningful set of confidence for the time-varying matrices $P_t^a$. In this paper we will use the first model of uncertainty in order to derive statistically meaningful sets of confidence for the transition matrices, based on likelihood or entropy bounds. Then, instead of solving the corresponding difficult control problem (3), we use an approximation that is common in robust control, and solve the time-varying upper bound (4), using the uncertainty sets $\mathcal{P}^a$ derived from a stationarity assumption about the transition matrices. We will also consider a variant of the finite-horizon time-varying problem (4), where controller and nature play alternatively, leading to a repeated game

$$\phi_N^{\mathrm{rep}}(\Pi, \mathcal{Q}) := \min_{\mathbf{a}_0} \max_{\tau_0 \in \mathcal{Q}} \min_{\mathbf{a}_1} \max_{\tau_1 \in \mathcal{Q}} \ldots \min_{\mathbf{a}_{N-1}} \max_{\tau_{N-1} \in \mathcal{Q}} C_N(\pi, \tau), \tag{5}$$

where the notation $\tau_t = (P_t^a)_{a \in \mathcal{A}}$ denotes the collection of transition matrices at a given time $t \in T$, and $\mathcal{Q} := \otimes_{a \in \mathcal{A}} \mathcal{P}^a$ is the corresponding set of confidence.

Finally, we will consider an infinite-horizon robust control problem, with the discounted cost function referred to above, and where we restrict control and nature policies to be stationary:

$$\phi_\infty(\Pi_s, \mathcal{T}_s) := \min_{\pi \in \Pi_s} \max_{\tau \in \mathcal{T}_s} C_\infty(\pi, \tau), \tag{6}$$

where $\Pi_s$ denotes the space of stationary control policies. We define $\phi_\infty(\Pi, \mathcal{T})$, $\phi_\infty(\Pi, \mathcal{T}_s)$ and $\phi_\infty(\Pi_s, \mathcal{T})$ accordingly.

In the sequel, for a given control policy $\pi \in \Pi$ and subset $\mathcal{S} \subseteq \mathcal{T}$, the notation $\phi_N(\pi, \mathcal{S}) := \max_{\tau \in \mathcal{S}} C_N(\pi, \tau)$ denotes the worst-case expected total cost for the finite-horizon problem, and $\phi_\infty(\pi, \mathcal{S})$ is defined likewise.

## 2.2  Main results

Our main contributions are as follows. First we provide a recursion, the "robust dynamic programming" algorithm, which solves the finite-horizon robust control problem (4). We

provide a simple proof in [2] of the optimality of the recursion, where the main ingredient is to show that perfect duality holds in the game (4). As a corollary of this result, we obtain that the repeated game (5) is equivalent to its non-repeated counterpart (4). Second, we provide similar results for the infinite-horizon problem with discounted cost function, (6). Moreover, we obtain that if we consider a finite-horizon problem with a discounted cost function, then the gap between the optimal value of the stationary uncertainty problem (3) and that of its time-varying counterpart (4) goes to zero as the horizon length goes to infinity, at a rate determined by the discount factor. Finally, we identify several classes of uncertainty models, which result in an algorithm that is *both* statistically accurate and numerically tractable. We provide precise complexity results that imply that, with the proposed approach, robustness can be handled at practically no extra computing cost.

## 3 Finite-Horizon Robust MDP

We consider the finite-horizon robust control problem defined in section 2.1. For a given state $i \in \mathcal{X}$, action $a \in \mathcal{A}$, and $P^a \in \mathcal{P}^a$, we denote by $p_i^a$ the next-state distribution drawn from $P^a$ corresponding to state $i \in \mathcal{X}$; thus $p_i^a$ is the $i$-th row of matrix $P^a$. We define $\mathcal{P}_i^a$ as the projection of the set $\mathcal{P}^a$ onto the set of $p_i^a$-variables. By assumption, these sets are included in the probability simplex of $\mathbf{R}^n$, $\Delta_n$; no other property is assumed. The following theorem is proved in [2].

**Theorem 1 (robust dynamic programming)** *For the robust control problem (4), perfect duality holds:*

$$\phi_N(\Pi, \mathcal{T}) = \min_{\pi \in \Pi} \max_{\tau \in \mathcal{T}} C_N(\pi, \tau) = \max_{\tau \in \mathcal{T}} \min_{\pi \in \Pi} C_N(\pi, \tau) := \psi_N(\Pi, \mathcal{T}).$$

*The problem can be solved via the recursion*

$$v_t(i) = \min_{a \in \mathcal{A}} \left( c_t(i, a) + \sigma_{\mathcal{P}_i^a}(v_{t+1}) \right), \quad i \in \mathcal{X}, \ t \in T, \tag{7}$$

*where $\sigma_{\mathcal{P}}(v) := \sup\{p^T v \ : \ p \in \mathcal{P}\}$ denotes the support function of a set $\mathcal{P}$, $v_t(i)$ is the worst-case optimal value function in state $i$ at stage $t$. A corresponding optimal control policy $\pi^* = (\mathbf{a}_0^*, \dots, \mathbf{a}_{N-1}^*)$ is obtained by setting*

$$\mathbf{a}_t^*(i) \in \arg\min_{a \in \mathcal{A}} \ \left\{ c_t(i, a) + \sigma_{\mathcal{P}_i^a}(v_{t+1}) \right\}, \quad i \in \mathcal{X}. \tag{8}$$

*The effect of uncertainty on a* given *strategy $\pi = (\mathbf{a}_0, \dots, \mathbf{a}_N)$ can be evaluated by the following recursion*

$$v_t^\pi(i) \quad = \quad c_t(i, \mathbf{a}_t(i)) + \sigma_{\mathcal{P}_i^{\mathbf{a}_t(i)}}(v_{t+1}^\pi), \quad i \in \mathcal{X}, \tag{9}$$

*which provides the worst-case value function $v^\pi$ for the strategy $\pi$.*

The above result has a nice consequence for the repeated game (5):

**Corollary 2** *The repeated game (5) is equivalent to the game (4):*

$$\phi_N^{\mathrm{rep}}(\Pi, \mathcal{Q}) = \phi_N(\Pi, \mathcal{T}),$$

*and the optimal strategies for $\phi_N(\Pi, \mathcal{T})$ given in theorem 1 are optimal for $\phi_N^{\mathrm{rep}}(\Pi, \mathcal{Q})$ as well.*

The interpretation of the perfect duality result given in theorem 1, and its consequence given in corollary 2, is that it does not matter wether the controller or nature play first, or if they alternatively; all these games are equivalent.

Each step of the robust dynamic programming algorithm involves the solution of an optimization problem, referred to as the "inner problem", of the form

$$\sigma_{\mathcal{P}_i^a}(v) = \max_{p \in \mathcal{P}_i^a} v^T p, \tag{10}$$

where $\mathcal{P}_i^a$ is the set that describes the uncertainty on $i$-th row of the transition matrix $P^a$, and $v$ contains the elements of the value function at some given stage. The complexity of the sets $\mathcal{P}_i^a$ for each $i \in \mathcal{X}$ and $a \in \mathcal{A}$ is a key component in the complexity of the robust dynamic programming algorithm. Beyond numerical tractability, an additional criteria for the choice of a specific uncertainty model is of course be that the sets $\mathcal{P}^a$ should represent accurate (non-conservative) descriptions of the statistical uncertainty on the transition matrices. Perhaps surprisingly, there are statistical models of uncertainty, such as those described in section 5, that are good on both counts. Precisely, these models result in inner problems (10) that can be solved in worst-case time of $O(n \log(v_{\max}/\delta))$ via a simple bisection algorithm, where $n$ is the size of the state space, $v_{\max}$ is a global upper bound on the value function, and $\delta > 0$ specifies the accuracy at which the optimal value of the inner problem (10) is computed. In the finite-horizon case, we can bound $v_{\max}$ by $O(N)$.

Now consider the following algorithm, where the uncertainty is described in terms of one of the models described in section 5:

**Robust Finite Horizon Dynamic Programming Algorithm**

1. *Set $\epsilon > 0$. Initialize the value function to its terminal value $\hat{v}_N = c_N$.*

2. *Repeat until $t = 0$:*

   (a) *For every state $i \in \mathcal{X}$ and action $a \in \mathcal{A}$, compute, using the bisection algorithm given in [2], a value $\hat{\sigma}_i^a$ such that*

   $$\hat{\sigma}_i^a - \epsilon/N \leq \sigma_{\mathcal{P}_i^a}(\hat{v}_t) \leq \hat{\sigma}_i^a.$$

   (b) *Update the value function by $\hat{v}_{t-1}(i) = \min_{a \in \mathcal{A}}(c_{t-1}(i,a) + \hat{\sigma}_i^a), i \in \mathcal{X}$.*
   (c) *Replace $t$ by $t-1$ and go to 2.*

3. *For every $i \in \mathcal{X}$ and $t \in T$, set $\pi^\epsilon = (\mathbf{a}_0^\epsilon, \ldots, \mathbf{a}_{N-1}^\epsilon)$, where*

   $$\mathbf{a}_t^\epsilon(i) = \arg\max_{a \in \mathcal{A}} \{c_{t-1}(i,a) + \hat{\sigma}_i^a\}, \quad i \in \mathcal{X}, \ a \in \mathcal{A}.$$

As shown in [2], the above algorithm provides an suboptimal policy $\pi^\epsilon$ that achieves the exact optimum with prescribed accuracy $\epsilon$, with a required number of flops bounded above by $O(mnN \log(N/\epsilon))$. This means that robustness is obtained at a relative increase of computational cost of only $\log(N/\epsilon)$ with respect to the classical dynamic programming algorithm, which is small for moderate values of $N$. If $N$ is very large, we can turn instead to the infinite-horizon problem examined in section 4, and similar complexity results hold.

## 4 Infinite-Horizon MDP

In this section, we address a the infinite-horizon robust control problem, with a discounted cost function of the form (2), where the terminal cost is zero, and $c_t(i,a) = \nu^t c(i,a)$, where $c(i,a)$ is now a constant cost function, which we assume non-negative and finite everywhere, and $\nu \in (0,1)$ is a discount factor.

We begin with the infinite-horizon problem involving stationary control and nature policies defined in (6). The following theorem is proved in [2].

**Theorem 3 (Robust Bellman recursion)** *For the infinite-horizon robust control problem (6) with stationary uncertainty on the transition matrices, stationary control policies, and a discounted cost function with discount factor $\nu \in [0, 1)$, perfect duality holds:*

$$\phi_\infty(\Pi_s, \mathcal{T}_s) = \max_{\tau \in \mathcal{T}_s} \min_{\pi \in \Pi_s} C_\infty(\pi, \tau) := \psi_\infty(\Pi_s, \mathcal{T}_s). \tag{11}$$

*The optimal value is given by $\phi_\infty(\Pi_s, \mathcal{T}_s) = v(i_0)$, where $i_0$ is the initial state, and where the value function $v$ satisfies is the optimality conditions*

$$v(i) = \min_{a \in \mathcal{A}} \left( c(i, a) + \nu \sigma_{\mathcal{P}_i^a}(v) \right), \quad i \in \mathcal{X}. \tag{12}$$

*The value function is the unique limit value of the convergent vector sequence defined by*

$$v_{k+1}(i) = \min_{a \in \mathcal{A}} \left( c(i, a) + \nu \sigma_{\mathcal{P}_i^a}(v_k) \right), \quad i \in \mathcal{X}, \quad k = 1, 2, \ldots \tag{13}$$

*A stationary, optimal control policy $\pi = (\mathbf{a}^*, \mathbf{a}^*, \ldots)$ is obtained as*

$$\mathbf{a}^*(i) \in \arg \min_{a \in \mathcal{A}} \left\{ c(i, a) + \nu \sigma_{\mathcal{P}_i^a}(v) \right\}, \quad i \in \mathcal{X}. \tag{14}$$

Note that the problem of computing the dual quantity $\psi_\infty(\Pi_s, \mathcal{T}_s)$ given in (11), has been addressed in [1], where the authors provide the recursion (13) without proof.

Theorem (3) leads to the following corollary, also proved in [2].

**Corollary 4** *In the infinite-horizon problem, we can without loss of generality assume that the control and nature policies are stationary, that is,*

$$\phi_\infty(\Pi, \mathcal{T}) = \phi_\infty(\Pi_s, \mathcal{T}_s) = \phi_\infty(\Pi_s, \mathcal{T}) = \phi_\infty(\Pi, \mathcal{T}_s). \tag{15}$$

*Furthermore, in the finite-horizon case, with a discounted cost function, the gap between the optimal values of the finite-horizon problems under stationary and time-varying uncertainty models, $\phi_N(\Pi, \mathcal{T}) - \phi_N(\Pi, \mathcal{T}_s)$, goes to zero as the horizon length $N$ goes to infinity, at a geometric rate $\nu$.*

Now consider the following algorithm, where we describe the uncertainty using one of the models of section 5.

### Robust Infinite Horizon Dynamic Programming Algorithm

1. *Set $\epsilon > 0$, initialize the value function $\hat{v}_1 > 0$ and set $k = 1$.*

2. (a) *For all states $i$ and controls $a$, compute, using the bisection algorithm given in [2], a value $\hat{\sigma}_i^a$ such that*

   $$\hat{\sigma}_i^a - \delta \leq \sigma_{\mathcal{P}_i^a}(\hat{v}_k) \leq \hat{\sigma}_i^a,$$

   *where $\delta = (1 - \nu)\epsilon/2\nu$.*
   (b) *For all states $i$ and controls $a$, compute $\hat{v}_{k+1}(i)$ by,*

   $$\hat{v}_{k+1}(i) = \min_{a \in \mathcal{A}} \left( c(i, a) + \nu \hat{\sigma}_i^a \right).$$

3. *If*

   $$\| \hat{v}_{k+1} - \hat{v}_k \| < \frac{(1 - \nu)\epsilon}{2\nu},$$

   *go to 4. Otherwise, replace $k$ by $k + 1$ and go to 2.*

4. *For each $i \in \mathcal{X}$, set an $\pi^\epsilon = (\mathbf{a}^\epsilon, \mathbf{a}^\epsilon, \ldots)$, where*

   $$\mathbf{a}^\epsilon(i) = \arg \max_{a \in \mathcal{A}} \left\{ c(i, a) + \nu \hat{\sigma}_i^a \right\}, \quad i \in \mathcal{X}.$$

In [2], we establish that the above algorithm finds an $\epsilon$-suboptimal robust policy in at most $O(nm \log(1/\epsilon)^2)$ flops. Thus, the extra computational cost incurred by robustness in the infinite-horizon case is only $O(\log(1/\epsilon))$.

## 5 Kullback-Liebler Divergence Uncertainty Models

We now address the inner problem (10) for a specific action $a \in \mathcal{A}$ and state $i \in \mathcal{X}$. Denote by $D(p\|q)$ denotes the Kullback-Leibler (KL) divergence (relative entropy) from the probability distribution $q \in \Delta_n$ to the probability distribution $p \in \Delta_n$:

$$D(p\|q) := \sum_j p(j) \log \frac{p(j)}{q(j)}.$$

The above function provides a natural way to describe errors in (rows of) the transition matrices; examples of models based on this function are given below.

*Likelihood Models:* Our first uncertainty model is derived from a controlled experiment starting from state $i = 1, 2, \ldots, n$ and the count of the number of transitions to different states. We denote by $F^a$ the matrix of empirical frequencies of transition with control $a$ in the experiment; denote by $f_i^a$ its $i^{\text{th}}$ row. We have $F^a \geq 0$ and $F^a \mathbf{1} = \mathbf{1}$, where $\mathbf{1}$ denotes the vector of ones. The "plug-in" estimate $\hat{P}^a = F^a$ is the solution to the maximum likelihood problem

$$\max_P \sum_{i,j} F^a(i,j) \log P(i,j) \; : \; P \geq 0, \;\; P\mathbf{1} = \mathbf{1}. \tag{16}$$

The optimal log-likelihood is $\beta_{\max}^a = \sum_{i,j} F^a(i,j) \log F^a(i,j)$. A classical description of uncertainty in a maximum-likelihood setting is via the "likelihood region" [6]

$$\mathcal{P}^a = \left\{ P \in \mathbf{R}^{n \times n} \; : \; P \geq 0, \;\; P\mathbf{1} = \mathbf{1}, \;\; \sum_{i,j} F^a(i,j) \log P(i,j) \geq \beta^a \right\}, \tag{17}$$

where $\beta^a < \beta_{\max}^a$ is a pre-specified number, which represents the uncertainty level. In practice, the designer specifies an uncertainty level $\beta^a$ based on re-sampling Bmethods, or on a large-sample Gaussian approximation, so as to ensure that the set above achieves a desired level of confidence.

With the above model, we note that the inner problem (10) only involves the set $\mathcal{P}_i^a := \left\{ p_i^a \in \mathbf{R}^n \; : \; p_i^a \geq 0, \;\; p_i^{aT}\mathbf{1} = 1, \;\; \sum_j F^a(i,j) \log p_i^a(j) \geq \beta_i^a \right\}$, where the parameter $\beta_i^a := \beta^a - \sum_{k \neq i} \sum_j F^a(k,j) \log F^a(k,j)$. The set $\mathcal{P}_i^a$ is the *projection* of the set described in (17) on a specific axis of $p_i^a$-variables. Noting further that the likelihood function can be expressed in terms of KL divergence, the corresponding uncertainty model on the row $p_i^a$ for given $i \in \mathcal{X}$, $a \in \mathcal{A}$, is given by a set of the form $\mathcal{P}_i^a = \{ p \in \Delta_n \; : \; D(f_i^a\|p) \leq \gamma_i^a \}$, where $\gamma_i^a = \sum_j F^a(i,j) \log F^a(i,j) - \beta_i^a$ is a function of the uncertainty level.

*Maximum A-Posteriori (MAP) Models:* a variation on Likelihood models involves Maximum A Posteriori (MAP) estimates. If there exist a prior information regrading the uncertainty on the $i$-th row of $P^a$, which can be described via a Dirichlet distribution [4] with parameter $\alpha_i^a$, the resulting MAP estimation problem takes the form

$$\max_p (f_i^a + \alpha_i^a - \mathbf{1})^T \log p \; : \; p^T\mathbf{1} = 1, p \geq 0.$$

Thus, the MAP uncertainty model is equivalent to a Likelihood model, with the sample distribution $f_i^a$ replaced by $f_i^a + \alpha_i^a - \mathbf{1}$, where $\alpha_i^a$ is the prior corresponding to state $i$ and action $a$.

*Relative Entropy Models:* Likelihood or MAP models involve the KL divergence from the unknown distribution to a reference distribution. We can also choose to describe uncertainty by exchanging the order of the arguments of the KL divergence. This results in a

so-called "relative entropy" model, where the uncertainty on the $i$-th row of the transition matrix $P^a$ described by a set of the form $\mathcal{P}_i^a = \{p \in \Delta_n \; : \; D(p\|q_i^a) \leq \gamma_i^a\}$, where $\gamma_i^a > 0$ is fixed, $q_i^a > 0$ is a given "reference" distribution (for example, the Maximum Likelihood distribution).

Equipped with one of the above uncertainty models, we can address the inner problem (10). As shown in [2], the inner problem can be converted by convex duality, to a problem of minimizing a single-variable, convex function. In turn, this one-dimensional convex optimization problem can be solved via a bisection algorithm with a worst-case complexity of $O(n\log(v_{\max}/\delta))$, where $\delta > 0$ specifies the accuracy at which the optimal value of the inner problem (10) is computed, and $v_{\max}$ is a global upper bound on the value function.

*Remark:* We can also use models where the uncertainty in the $i$-th row for the transition matrix $P^a$ is described by a finite set of vectors, $\mathcal{P}_i^a = \{p_i^{a,1}, \ldots, p_i^{a,K}\}$. In this case the complexity of the corresponding robust dynamic programming algorithm is increased by a relative factor of $K$ with respect to its classical counterpart, which makes the approach attractive when the number of "scenarios" $K$ is moderate.

## 6 Concluding remarks

We proposed a "robust dynamic programming" algorithm for solving finite-state and finite-action MDPs whose solutions are guaranteed to tolerate arbitrary changes of the transition probability matrices within given sets. We proposed models based on KL divergence, which is a natural way to describe estimation errors. The resulting robust dynamic programming algorithm has almost the same computational cost as the classical dynamic programming algorithm: the relative increase to compute an $\epsilon$-suboptimal policy is $O(N\log(1/\epsilon))$ in the $N$-horizon case, and $O(\log(1/\epsilon))$ for the infinite-horizon case.

## Footnotes

*Research funded in part by Eurocontrol-014692, DARPA-F33615-01-C-3150, and NSF-ECS-9983874

## References

[1] J. Bagnell, A. Ng, and J. Schneider. Solving uncertain Markov decision problems. Technical Report CMU-RI-TR-01-25, Robotics Institute, Carnegie Mellon University, August 2001.

[2] L. El-Ghaoui and A. Nilim. Robust solution to Markov decision problems with uncertain transition matrices: proofs and complexity analysis. Technical Report UCB/ERL M04/07, Department of EECS, University of California, Berkeley, January 2004. A related version has been submitted to *Operations Research* in Dec. 2003.

[3] E. Feinberg and A. Shwartz. *Handbook of Markov Decision Processes, Methods and Applications*. Kluwer's Academic Publishers, Boston, 2002.

[4] T. Ferguson. Prior distributions on space of probability measures. *The Annal of Statistics*, 2(4):615–629, 1974.

[5] R. Givan, S. Leach, and T. Dean. Bounded parameter Markov decision processes. In *fourth European Conference on Planning*, pages 234–246, 1997.

[6] E. Lehmann and G. Casella. *Theory of point estimation*. Springer-Verlag, New York, USA, 1998.

[7] M. Putterman. *Markov Decision Processes: Discrete Stochastic Dynamic Programming*. Wiley-Interscince, New York, 1994.

[8] J. K. Satia and R. L. Lave. Markov decision processes with uncertain transition probabilities. *Operations Research*, 21(3):728–740, 1973.

[9] A. Shapiro and A. J. Kleywegt. Minimax analysis of stochastic problems. *Optimization Methods and Software*, 2002. to appear.

[10] C. C. White and H. K. Eldeib. Markov decision processes with imprecise transition probabilities. *Operations Research*, 42(4):739–749, 1994.
